# Probabilistic Deterministic Infinite Automata

**David Pfau**        **Nicholas Bartlett**        **Frank Wood**
Columbia University, New York, NY 10027, USA
{pfau@neurotheory,{bartlett,fwood}@stat}.columbia.edu

## Abstract

We propose a novel Bayesian nonparametric approach to learning with probabilistic deterministic finite automata (PDFA). We define and develop a sampler for a PDFA with an infinite number of states which we call the probabilistic deterministic infinite automata (PDIA). Posterior predictive inference in this model, given a finite training sequence, can be interpreted as averaging over multiple PDFAs of varying structure, where each PDFA is biased towards having few states. We suggest that our method for averaging over PDFAs is a novel approach to predictive distribution smoothing. We test PDIA inference both on PDFA structure learning and on both natural language and DNA data prediction tasks. The results suggest that the PDIA presents an attractive compromise between the computational cost of hidden Markov models and the storage requirements of hierarchically smoothed Markov models.

## 1  Introduction

The focus of this paper is a novel Bayesian framework for learning with probabilistic deterministic finite automata (PDFA) [9]. A PDFA is a generative model for sequential data (PDFAs are reviewed in Section 2). Intuitively a PDFA is similar to a hidden Markov model (HMM) [10] in that it consists of a set of states, each of which when visited emits a symbol according to an emission probability distribution. It differs from an HMM in how state-to-state transitions occur; transitions are deterministic in a PDFA and nondeterministic in an HMM.

In our framework for learning with PDFAs we specify a prior over the parameters of a single large PDFA that encourages state reuse. The inductive bias introduced by the PDFA prior provides a soft constraint on the number of states used to generate the data. We take the limit as the number of states becomes infinite, yielding a model we call the probabilistic deterministic infinite automata (PDIA).

Given a finite training sequence, the PDIA posterior distribution is an infinite mixture of PDFAs. Samples from this distribution form a finite sample approximation to this infinite mixture, and can be drawn via Markov chain Monte Carlo (MCMC) [6]. Using such a mixture we can average over our uncertainty about the model parameters (including state cardinality) in a Bayesian way during prediction and other inference tasks. We find that averaging over a finite number of PDFAs trained on naturalistic data leads to better predictive performance than using a single "best" PDFA.

We chose to investigate learning with PDFAs because they are intermediate in expressive power between HMMs and finite-order Markov models, and thus strike a good balance between generalization performance and computational efficiency. A single PDFA is known to have relatively limited expressivity. We argue that a finite mixture of PDFAs has greater expressivity than that of a single PDFA but is not as expressive as a probabilistic nondeterministic finite automata (PNFA)[1]. A PDIA is clearly highly expressive; an infinite mixture over the same is even more so. Even though ours is a Bayesian approach to PDIA learning, in practice we only ever deal with a finite approximation to the full posterior and thus limit our discussion to finite mixtures of PDFAs.

While model expressivity is a concern, computational considerations often dominate model choice. We show that prediction in a trained mixture of PDFAs can have lower asymptotic cost than forward prediction in the PNFA/HMM class of models. We also present evidence that averaging over PDFAs gives predictive performance superior to HMMs trained with standard methods on naturalistic data. We find that PDIA predictive performance is competitive with that of fixed-order, smoothed Markov models with the same number of states. While sequence learning approaches such as the HMM and smoothed Markov models are well known and now highly optimized, our PDIA approach to learning is novel and is amenable to future improvement.

Section 2 reviews PDFAs, Section 3 introduces Bayesian PDFA inference, Section 4 presents experimental results on DNA and natural language, and Section 5 discusses related work on PDFA induction and the theoretical expressive power of mixtures of PDFAs. In Section 6 we discuss ways in which PDIA predictive performance might be improved in future research.

## 2  Probabilistic Deterministic Finite Automata

A PDFA is formally defined as a 5-tuple $M = (Q, \Sigma, \delta, \pi, q_0)$, where $Q$ is a finite set of states, $\Sigma$ is a finite alphabet of observable symbols, $\delta : Q \times \Sigma \to Q$ is the transition function from a state/symbol pair to the next state, $\pi : Q \times \Sigma \to [0, 1]$ is the probability of the next symbol given a state and $q_0$ is the initial state.[2] Throughout this paper we will use $i$ to index elements of $Q$, $j$ to index elements of $\Sigma$, and $t$ to index elements of an observed string. For example, $\delta_{ij}$ is shorthand for $\delta(q_i, \sigma_j)$, where $q_i \in Q$ and $\sigma_j \in \Sigma$.

Given a state $q_i$, the probability that the next symbol takes the value $\sigma_j$ is given by $\pi(q_i, \sigma_j)$. We use the shorthand $\boldsymbol{\pi}_{q_i}$ for the state-specific discrete distribution over symbols for state $q_i$. We can also write $\sigma | q_i \sim \boldsymbol{\pi}_{q_i}$ where $\sigma$ is a random variable that takes values in $\Sigma$. Given a state $q_i$ and a symbol $\sigma_j$, however, the next state $q_{i'}$ is *deterministic*: $q_{i'} = \delta(q_i, \sigma_j)$. Generating from a PDFA involves first generating a symbol stochastically given the state the process is in: $x_t | \xi_t \sim \boldsymbol{\pi}_{\xi_t}$ where $\xi_t \in Q$ is the state at time $t$. Next, given $\xi_t$ and $x_t$ transitioning deterministically to the next state: $\xi_{t+1} = \delta(\xi_t, x_t)$. This is the reason for the confusing "probabilistic deterministic" name for these models. Turning this around, given data, $q_0$, and $\delta$, there is no uncertainty about the path through the states. This is a primary source of computational savings relative to HMMs.

PDFAs are more general than $n$th-order Markov models (i.e. $m$-gram models, $m = n + 1$), but less expressive than hidden Markov models (HMMs)[3]. For the case of $n$th-order Markov models, we can construct a PDFA with one state per suffix $x_1 x_2 \ldots x_n$. Given a state and a symbol $x_{n+1}$, the unique next state is the one corresponding to the suffix $x_2 \ldots x_{n+1}$. Thus $n$th-order Markov models are a subclass of PDFAs with $\mathcal{O}(|\Sigma|^n)$ states. For an HMM, given data and an initial distribution over states, there is a posterior probability for every path through the state space. PDFAs are those HMMs for which, given a unique start state, the posterior probability over paths is degenerate at a single path. As we explain in Section 5, mixtures of PDFAs are strictly more expressive than single PDFAs, but still less expressive than PNFAs.

## 3  Bayesian PDFA Inference

We start our description of Bayesian PDFA inference by defining a prior distribution over the parameters of a finite PDFA. We then show how to analytically marginalize nuisance parameters out of the model and derive a Metropolis-Hastings sampler for posterior inference using the resulting collapsed representation. We discuss the limit of our model as the number of states in the PDFA goes to infinity. We call this limit the probabilistic deterministic infinite automaton (PDIA). We develop a PDIA sampler that carries over from the finite case in a natural way.

### 3.1  A PDFA Prior

We assume that the set of states $Q$, set of symbols $\Sigma$, and initial state $q_0$ of a PDFA are known but that the transition and emission functions are unknown. The PDFA prior then consists of a prior over both the transition function $\delta$ and the emission probability function $\pi$. In the finite case $\delta$ and

$\pi$ are representable as finite matrices, with one column per element of $\Sigma$ and one row per element of $Q$. For each column $j$ ($j$ co-indexes columns and set elements) of the transition matrix $\delta$, our prior stipulates that the elements of that column are i.i.d. draws from a discrete distribution $\phi_j$ over $Q$, that is, $\delta_{ij} \sim [\phi_1, \ldots, \phi_{|\Sigma|}]$, $0 \le i \le |Q| - 1$. The $\phi_j$ represent transition tendencies given a symbol, if the $i$th element of $\phi_j$ is large then state $q_i$ is likely to be transitioned to anytime the last symbol was $\sigma_j$. The $\phi_j$'s are themselves given a shared Dirichlet prior with parameters $\alpha\mu$, where $\alpha$ is a concentration and $\mu$ is a template transition probability vector. If the $i$th element of $\mu$ is large then the $i$th state is likely to be transitioned to regardless of the emitted symbol. We place a uniform Dirichlet prior on $\mu$ itself, with $\gamma$ total mass and average over $\mu$ during inference. This hierarchical Dirichlet construction encourages both general and context specific state reuse. We also place a uniform Dirichlet prior over the per-state emission probabilities $\pi_{q_i}$ with $\beta$ total mass which smooths emission distribution estimates. Formally:

$$
\begin{aligned}
\mu | \gamma, |Q| &\sim \mathrm{Dir}\left(\gamma/|Q|, \ldots, \gamma/|Q|\right) & (1) \\
\phi_j | \alpha, \mu &\sim \mathrm{Dir}(\alpha\mu) & (2) \\
\pi_{q_i} | \beta, |\Sigma| &\sim \mathrm{Dir}(\beta/|\Sigma|, \ldots, \beta/|\Sigma|) \\
\delta_{ij} &\sim \phi_j
\end{aligned}
$$

where $0 \le i \le |Q| - 1$ and $1 \le j \le |\Sigma|$. Given a sample from this model we can run the PDFA to generate a sequence of $T$ symbols. Using $\xi_t$ to denote the state of the PDFA at position $t$ in the sequence:

$$
\xi_0 = q_0, \qquad x_0 \sim \pi_{q_0}, \qquad \xi_t = \delta(\xi_{t-1}, x_{t-1}), \qquad x_t \sim \pi_{\xi_t}
$$

We choose this particular inductive bias, with transitions tied together within a column of $\delta$, because we wanted the most recent symbol emission to be informative about what the next state is. If we instead had a single Dirichlet prior over all elements of $\delta$, transitions to a few states would be highly likely no matter the context and those states would dominate the behavior of the automata. If we tied together rows of $\delta$ instead of columns, being in a particular state would tell us more about the sequence of states we came from than the symbols that got us there.

Note that this prior stipulates a fully connected PDFA in which all states may transition to all others and all symbols may be emitted from each state. This is slightly different that the canonical finite state machine literature where sparse connectivity is usually the norm.

### 3.2 PDFA Inference

Given observational data, we are interested in learning a posterior distribution over PDFAs. We do this by GIbbs sampling the transition matrix $\delta$ with $\pi$ and $\phi_j$ integrated out. To start inference we need the likelihood function for a fixed PDFA; it is given by

$$
p(x_{0:T} | \pi, \delta) = \pi(\xi_0, x_0) \prod_{t=1}^{T} \pi(\xi_t, x_t).
$$

Remember that $\xi_t | \xi_{t-1}, x_{t-1}$ is deterministic given the transition function $\delta$. We can marginalize $\pi$ out of this expression and express the likelihood of the data in a form that depends only on the counts of symbols emitted from each state. Define the count matrix $c$ for the sequence $x_{0:T}$ and transition matrix $\delta$ as $c_{ij} = \sum_{t=0}^{T} I_{ij}(\xi_t, x_t)$, where $I_{ij}(\xi_t, x_t)$ is an indicator function for the automaton being in state $q_i$ when it generates $x_t$, i.e. $\xi_t = q_i$ and $x_t = \sigma_j$. This matrix $c = [c_{ij}]$ gives the number of times each symbol is emitted from each state. Due to multinomial-Dirichlet conjugacy we can express the probability of a sequence given the transition function $\delta$, the count matrix $c$ and $\beta$:

$$
p(x_{0:T} | \delta, c, \beta) = \int p(x_{0:T} | \pi, \delta) p(\pi | \beta) d\pi = \prod_{i=0}^{|Q|-1} \frac{\Gamma(\beta)}{\Gamma(\frac{\beta}{|\Sigma|})^{|\Sigma|}} \frac{\prod_{j=1}^{|\Sigma|} \Gamma(\frac{\beta}{|\Sigma|} + c_{ij})}{\Gamma(\beta + \sum_{j=1}^{|\Sigma|} c_{ij})} \quad (3)
$$

If the transition matrix $\delta$ is observed we have a closed-form expression for its likelihood given $\mu$ with all $\phi_j$'s marginalized out. Let $v_{ij}$ be the number of times state $q_i$ is transitioned to given that $\sigma_j$ was the last symbol emitted, i.e. $v_{ij}$ is the number of times $\delta_{i'j} = q_i$ for all states $i'$ in the column

$j$. The marginal likelihood of $\delta$ in terms of $\boldsymbol{\mu}$ is then:

$$p(\delta|\boldsymbol{\mu},\alpha) \quad = \quad \int p(\delta|\phi)p(\phi|\boldsymbol{\mu},\alpha)d\phi = \prod_{j=1}^{|\Sigma|} \frac{\Gamma(\alpha)}{\prod_{i=0}^{|Q|-1}\Gamma(\alpha\mu_i)} \frac{\prod_{i=0}^{|Q|-1}\Gamma(\alpha\mu_i + v_{ij})}{\Gamma(\alpha + |Q|)} \tag{4}$$

We perform posterior inference in the finite model by sampling elements of $\delta$ and the vector $\boldsymbol{\mu}$. One can sample $\delta_{ij}$ given the rest of the matrix $\delta_{-ij}$ using

$$p(\delta_{ij}|\delta_{-ij}, x_{0:T}, \boldsymbol{\mu}, \alpha) \propto p(x_{0:T}|\delta_{ij}, \delta_{-ij})p(\delta_{ij}|\delta_{-ij}, \boldsymbol{\mu}, \alpha) \tag{5}$$

Both terms on the right hand side of this equation have closed-form expressions, the first given in (3). The second can be found from (4) and is

$$P(\delta_{ij} = q_{i'}|\delta_{-ij}, \alpha, \boldsymbol{\mu}) = \frac{\alpha\mu_{i'} + v_{i'j}}{\alpha + |Q| - 1} \tag{6}$$

where $v_{i'j}$ is the number of elements in column $j$ equal to $q_{i'}$ excluding $\delta_{ij}$. As $|Q|$ is finite, we compute (5) for all values of $\delta_{ij}$ and normalize to produce the required conditional probability distribution.

Note that in (3), the count matrix $c$ may be profoundly impacted by changing even a single element of $\delta$. The values in $c$ depend on the specific sequence of states the automata used to generate $x$. Changing the value of a single element of $\delta$ affects the state trajectory the PDFA must follow to generate $x_{0:T}$. Among other things this means that some elements of $c$ that were nonzero may become zero, and vice versa.

We can reduce the computational cost of inference by deleting transitions $\delta_{ij}$ for which the corresponding counts $c_{ij}$ become 0. In practical sampler implementations this means that one need not even represent transitions corresponding to zero counts. The likelihood of the data (3) does not depend on the value of $\delta_{ij}$ if symbol $\sigma_j$ is never emitted while the machine is in state $q_i$. In this case sampling from (5) is the same as sampling without conditioning on the data at all. Thus, if while sampling we change some transition that renders $c_{ij} = 0$ for some values for each of $i$ and $j$, we can delete $\delta_{ij}$ until another transition is changed such that $c_{ij}$ becomes nonzero again, when we sample $\delta_{ij}$ anew. Under the marginal joint distribution of a column of $\delta$ the row entries in that column are exchangeable, and so deleting an entry of $\delta$ has the same effect as marginalizing it out. When all $\delta_{ij}$ for some state $q_i$ are marginalized out, we can say the state itself is marginalized out. When we delete an element from a column of $\delta$, we replace the $|Q| - 1$ in the denominator of (6) with $D_j^+ = \sum_{i=0}^{|Q|-1} I(v_{ij} \neq 0)$, the number of entries in the $j$th column of $\delta$ that are *not* marginalized out yielding

$$P(\delta_{ij} = q_{i'}|\delta_{-ij}, \alpha, \boldsymbol{\mu}) = \frac{\alpha\mu_{i'} + v_{i'j}}{\alpha + D_j^+}. \tag{7}$$

If when sampling $\delta_{ij}$ it is assigned it a state $q_{i'}$ such that some $c_{i'j'}$ which was zero is now nonzero, we simply reinstantiate $\delta_{i'j'}$ by drawing from (7) and update $D_{j'}^+$. When sampling a single $\delta_{ij}$ there can be many such transitions as the path through the machine dictated by $x_{0:T}$ may use many transitions in $\delta$ that were deleted. In this case we update incrementally, increasing $D_j^+$ and $v_{ij}$ as we go.

While it is possible to construct a Gibbs sampler using (5) in this collapsed representation, such a sampler requires a Monte Carlo integration over a potentially large subset of the marginalized-out transitions in $\delta$, which may be costly. A simpler strategy is to pretend that all entries of $\delta$ exist but are sampled in a "just-in-time" manner. This gives rise to a Metropolis Hastings (MH) sampler for $\delta$ where the proposed value for $\delta_{ij}$ is either one of the instantiated states or any one of the equivalent marginalized out states. Any time any marginalized out element of $\delta$ is required we can pretend as if we had just sampled its value, and we know that because its value had no effect on the likelihood of the data, we know that it would have been sampled directly from (7). It is in this sense that all marginalized out states are equivalent – we known nothing more about their connectivity structure than that given by the prior in (7).

For the MH sampler, denote the set of non-marginalized out $\delta$ entries $\delta^+ = \{\delta_{ij} : c_{ij} > 0\}$. We propose a new value $q_{i*}$ for one $\delta_{ij} \in \delta^+$ according to (7). The conditional posterior probability

|      | PDIA  | PDIA-MAP | HMM-EM | bigram | trigram | 4-gram | 5-gram | 6-gram | SSM     |
|------|-------|----------|--------|--------|---------|--------|--------|--------|---------|
| AIW  | 5.13  | 5.46     | 7.89   | 9.71   | 6.45    | 5.13   | 4.80   | 4.69   | 4.78    |
|      | 365.6 | 379      | 52     | 28     | 382     | 2,023  | 5,592  | 10,838 | 19,358  |
| DNA  | 3.72  | 3.72     | 3.76   | 3.77   | 3.75    | 3.74   | 3.73   | 3.72   | 3.56    |
|      | 64.7  | 54       | 19     | 5      | 21      | 85     | 341    | 1,365  | 314,166 |

Table 1: PDIA inference performance relative to HMM and fixed order Markov models. Top rows: perplexity. Bottom rows: number of states in each model. For the PDIA this is an average number.

of this proposal is proportional to $p(x_{0:T}|\delta_{ij} = q_{i*}, \delta^+_{-ij})P(\delta_{ij} = q_{i*}|\delta^+_{-ij})$. The Hastings correction exactly cancels out the proposal probability in the accept/reject ratio leaving an MH accept probability for the $\delta_{ij}$ being set to $q_{i*}$ given that its previous value was $q_{i'}$ of

$$\alpha(\delta_{ij} = q_{i*}|\delta_{ij} = q_{i'}) = \min\left(1, \frac{p(x_{0:T}|\delta_{ij} = q_{i*}, \delta^+_{-ij})}{p(x_{0:T}|\delta_{ij} = q_{i'}, \delta^+_{-ij})}\right). \quad (8)$$

Whether $q_{i*}$ is marginalized out or not, evaluating $p(x_{0:T}|\delta_{ij} = q_{i*}, \delta^+_{-ij})$ may require reinstantiating marginalized out elements of $\delta$. As before, these values are sampled from (7) on a just-in-time schedule. If the new value is accepted, all $\delta_{ij} \in \delta^+$ for which $c_{ij} = 0$ are removed, and then move to the next transition in $\delta$ to sample.

In the finite case, one can sample $\boldsymbol{\mu}$ by Metropolis-Hastings or use a MAP estimate as in [7]. Hyperparameters $\alpha$, $\beta$ and $\gamma$ can be sampled via Metropolis-Hastings updates. In our experiments we use Gamma(1,1) hyperpriors.

### 3.3 The Probabilistic Deterministic Infinite Automaton

We would like to avoid placing a strict upper bound on the number of states so that model complexity can grow with the amount of training data. To see how to do this, consider what happens when $|Q| \to \infty$. In this case, the right hand side of equations (1) and (2) must be replaced by infinite dimensional alternatives

$$\begin{aligned}
\boldsymbol{\mu} &\sim \mathrm{PY}(\gamma, d_0, H) \\
\boldsymbol{\phi}_j &\sim \mathrm{PY}(\alpha, d, \boldsymbol{\mu}) \\
\delta_{ij} &\sim \boldsymbol{\phi}_j
\end{aligned}$$

where PY stands for Pitman Yor process and $H$ in our case is a geometric distribution over the integers with parameter $\lambda$. The resulting hierarchical model becomes the hierarchical Pitman-Yor process (HPYP) over a discrete alphabet [14]. The discount parameters $d_0$ and $d$ are particular to the infinite case, and when both are zero the HPYP becomes the well known hierarchical Dirichlet process (HDP), which is the infinite dimensional limit of (1) and (2) [15]. Given a finite amount of data, there can only be nonzero counts for a finite number of state/symbol pairs, so our marginalization procedure from the finite case will yield a $\delta$ with at most $T$ elements. Denote these non-marginalized out entries by $\delta^+$. We can sample the elements of $\delta^+$ as before using (8) provided that we can propose from the HPYP. In many HPYP sampler representations this is easy to do. We use the Chinese restaurant franchise representation [15] in which the posterior predictive distribution of $\delta_{ij}$ given $\delta^+_{-ij}$ can be expressed with $\boldsymbol{\phi}_j$ and $\boldsymbol{\mu}$ integrated out as

$$P(\delta_{ij} = q_{i'}|\delta^+_{-ij}, \alpha, \gamma) = \mathbb{E}\left[\frac{v_{i'j} - k_{i'j}d}{\alpha + D^+_j} + \frac{\alpha + k_{.j}d}{\alpha + D^+_j}\left(\frac{w_{i'} - \kappa_{i'}d_0}{\gamma + w.} + \frac{\gamma + \kappa.d_0}{\gamma + w.}H(q_{i'})\right)\right] \quad (9)$$

where $w_{i'}$, $k_{i'j}$, $\kappa_{i'}$, $w. = \sum_i w_i$, $k_{.j} = \sum_i k_{ij}$, and $\kappa. = \sum_i \kappa_i$ are stochastic bookkeeping counts required by the Chinese Restaurant franchise sampler. These counts must themselves be sampled [15]. The discount hyperparameters can also be sampled by Metropolis-Hastings.

## 4 Experiments and Results

To test our PDIA inference approach we evaluated it on discrete natural sequence prediction and compared its performance to HMMs and smoothed $n$-gram models. We trained the models on two

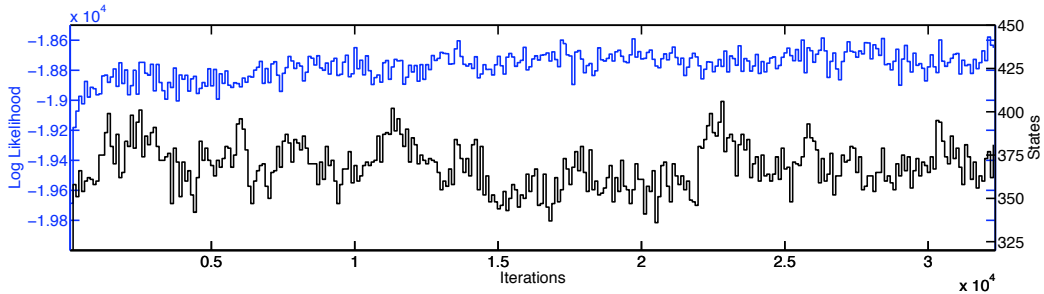

Figure 1: Subsampled PDIA sampler trace for Alice in Wonderland. The top trace is the joint log likelihood of the model and training data, the bottom trace is the number of states.

datasets: a character sequence from *Alice in Wonderland* [2] and a short sequence of mouse DNA. The *Alice in Wonderland* (AIW) dataset was preprocessed to remove all characters but letters and spaces, shift all letters from upper to lower case, and split along sentence dividers to yield a 27-character alphabet (a-z and space). We trained on 100 random sentences (9,986 characters) and tested on 50 random sentences (3,891 characters). The mouse DNA dataset consisted of a fragment of chromosome 2 with 194,173 base pairs, which we treated as a single unbroken string. We used the first 150,000 base pairs for training and the rest for testing. For AIW, the state of the PDIA model was always set to $q_0$ at the start of each sentence. For DNA, the state of the PDIA model at the start of the test data was set to the last state of the model after accepting the training data. We placed Gamma(1,1) priors over $\alpha$, $\beta$ and $\gamma$, set $\lambda = .001$, and used uniform priors for $d_0$ and $d$.

We evaluated the performance of the learned models by calculating the average per character predictive perplexity of the test data. For training data $x_{1:T}$ and test data $y_{1:T'}$ this is given by $2^{-\frac{1}{T'} \log_2 P(y_{1:T'}|x_{1:T})}$. It is a measure of the average uncertainty the model has about what character comes next given the sequence up to that point, and is at most $|\Sigma|$. We evaluated the probability of the test data incrementally, integrating the test data into the model in the standard Bayesian way.

Test perplexity results are shown in Table 1 on the first line of each subtable. Each sample passed through every instantiated transition. Every fifth sample for AIW and every tenth sample for DNA after burn-in was used for prediction. For AIW, we ran 15,000 burn-in samples and used 3,500 samples for predictive inference. Subsampled sampler diagnostic plots are shown in Figure 1 that demonstrate the convergence properties of our sampler. When modeling the DNA dataset we burn-in for 1,000 samples and use 900 samples for inference. For the smoothed $n$-gram models, we report thousand-sample average perplexity results for hierarchical Pitman-Yor process (HPYP) [14] models of varying Markov order (1 through 5 notated as bigram through 6-gram) after burning each model in for one hundred samples. We also show the performance of the single particle incremental variant of the sequence memoizer (SM) [5], the SM being the limit of an $n$-gram model as $n \to \infty$. We also show results for a hidden Markov model (HMM) [8] trained using expectation-maximization (EM). We determined the best number of hidden states by cross-validation on the test data (a procedure used here to produce optimistic HMM performance for comparison purposes only).

The performance of the PDIA exceeds that of the HMM and is approximately equal to that of a smoothed 4-gram model, though it does not outperform very deep, smoothed Markov models. This is in contrast to [16], which found that PDFAs trained on natural language data were able to predict as well as *unsmoothed* trigrams, but were significantly worse than smoothed trigrams, even when averaging over multiple learned PDFAs. As can be seen in the second line of each subtable in Table 1, the MAP number of states learned by the PDIA is significantly lower than that of the $n$-gram model with equal predictive performance.

Unlike the HMM, the computational complexity of PDFA prediction does not depend on the number of states in the model because only a single path through the states is followed. This means that the asymptotic cost of prediction for the PDIA is $\mathcal{O}(LT')$, where $L$ is the number of posterior samples and $T'$ is the length of the test sequence. For any single HMM it is $\mathcal{O}(KT')$, where $K$ is the number of states in the HMM. This is because all possible paths must be followed to achieve the given HMM predictive performance (although a subset of possible paths could be followed if doing approximate

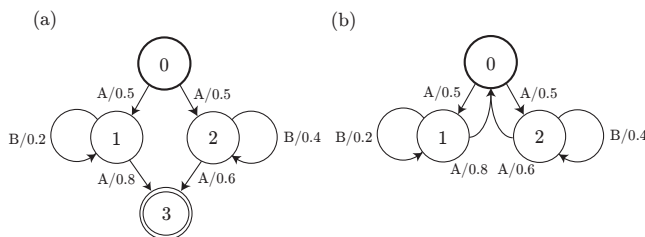

Figure 2: Two PNFAs outside the class of PDFAs. (a) can be represented by a mixture of two PDFAs, one following the right branch from state 0, the other following the left branch. (b), in contrast, cannot be represented by any finite mixture of PDFAs.

inference). In PDIA inference we too can choose the number of samples used for prediction, but here even a single sample has empirical prediction performance superior to averaging over all paths in an HMM. The computational complexity of smoothing $n$-gram inference is equivalent to PDIA inference, however, the storage cost for the large $n$-gram models is significantly higher than that of the estimated PDIA for the same predictive performance.

## 5   Theory and Related Work

The PDIA posterior distribution takes the form of an infinite mixture of PDFAs. In practice, we run a sampler for some number of iterations and approximate the posterior with a finite mixture of PDFAs. For this reason, we now consider the expressive power of finite mixtures of PDFAs. We show that they are strictly more expressive than PDFAs, but strictly less expressive than hidden Markov models. Probabilistic *non*-deterministic finite automata (PNFA) are a strictly larger model class than PDFAs. For example, the PNFA in 2(a) cannot be expressed as a PDFA [3]. However, it can be expressed as a mixture of two PDFAs, one with $Q = \{q_0, q_1, q_3\}$ and the other with $Q = \{q_0, q_2, q_3\}$. Thus mixtures of PDFAs are a strictly larger model class than PDFAs. In general, any PNFA where the nondeterministic transitions can only be visited once can be expressed as a mixture of PDFAs. However, if we replace transitions to $q_3$ with transitions to $q_0$, as in 2(b), there is no longer any equivalent finite mixture of PDFAs, since the nondeterministic branch from $q_0$ can be visited an arbitrary number of times.

Previous work on PDFA induction has focused on accurately discovering model structure when the true generative mechanism is a PDFA. State merging algorithms do this by starting with the trivial PDFA that only accepts the training data and merging states that pass a similarity test [1, 17], and have been proven to identify the correct model in the limit of infinite data. State splitting algorithms start at the opposite extreme, with the trivial single-state PDFA, and split states that pass a difference test [12, 13]. These algorithms return only a deterministic estimate, while ours naturally expresses uncertainty about the learned model.

To test if we can learn the generative mechanism given our inductive bias, we trained the PDIA on data from three synthetic grammars: the even process [13], the Reber grammar [11] and the Feldman grammar [4], which have up to 7 states and 7 symbols in the alphabet. In each case the mean number of states discovered by the model approached the correct number as more data was used in training. Results are presented in Figure 3. Furthermore, the predictive performance of the PDIA was nearly equivalent to the actual data generating mechanism.

## 6   Discussion

Our Bayesian approach to PDIA inference can be interpreted as a stochastic search procedure for PDFA structure learning where the number of states is unknown. In Section 5 we presented evidence that PDFA samples from our PDIA inference algorithm have the same characteristics as the true generative process. This in and of itself may be of interest to the PDFA induction community.

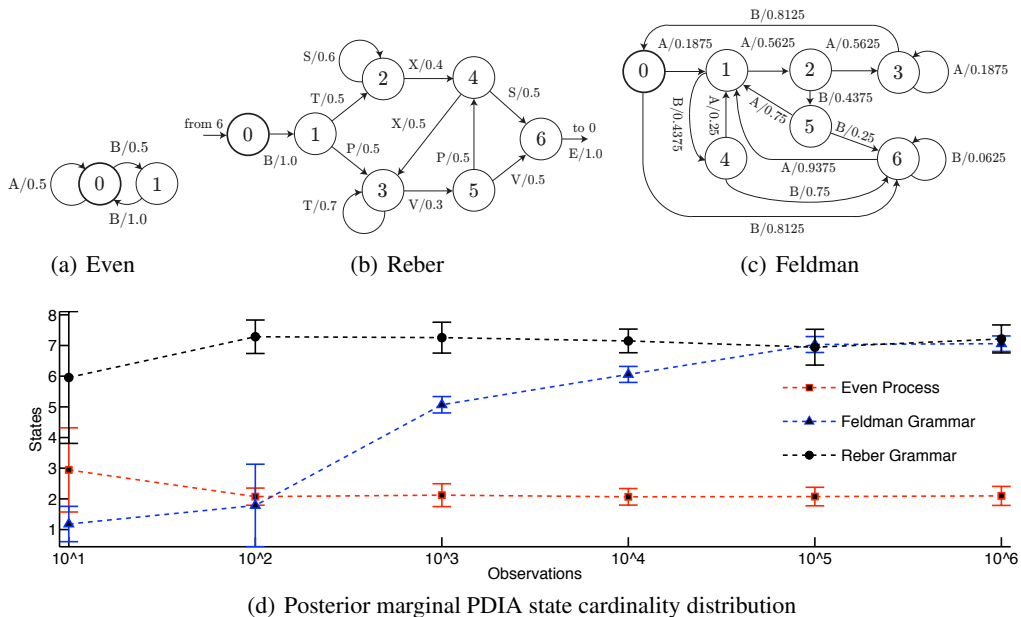

(a) Even        (b) Reber        (c) Feldman

(d) Posterior marginal PDIA state cardinality distribution

Figure 3: Three synthetic PDFAs: (a) even process [13], (b) Reber grammar [11], (c) Feldman grammar [4]. (d) posterior mean and standard deviation of number of states discovered during PDIA inference for varying amounts of data generated by each of the synthetic PDFAs. PDIA inference discovers PDFAs with the correct number of states

We ourselves are more interested in establishing new ways to produce smoothed predictive conditional distributions. Inference in the PDIA presents a completely new approach to smoothing, smoothing by averaging over PDFA model structure rather than hierarchically smoothing related emission distribution estimates. Our PDIA approach gives us an attractive ability to trade-off between model simplicity in terms of number of states, computational complexity in terms of asymptotic cost of prediction, and predictive perplexity. While our PDIA approach may not yet outperform the best smoothing Markov model approaches in terms of predictive perplexity alone, it does outperform them in terms of model complexity required to achieve the same predictive perplexity, and outperforms HMMs in terms of asymptotic time complexity of prediction. This suggests that a future combination of smoothing over model structure *and* smoothing over emission distributions could produce excellent results. PDIA inference gives researchers another tool to choose from when building models. If very fast prediction is desirable and the predictive perplexity difference between the PDIA and, for instance, the most competitive $n$-gram is insignificant from an application perspective, then doing finite sample inference in the PDIA offers a significant computational advantage in terms of memory.

We indeed believe the most promising approach to improving PDIA predictive performance is to construct a smoothing hierarchy over the state specific emission distributions, as is done in the smoothing $n$-gram models. For an $n$-gram, where every state corresponds to a suffix of the sequence, the predictive distributions for a suffix is smoothed by the predictive distribution for a shorter suffix, for which there are more observations. This makes it possible to increase the size of the model indefinitely without generalization performance suffering [18]. In the PDIA, by contrast, the predictive probabilities for states are not tied together. Since states of the PDIA are not uniquely identified by suffixes, it is no longer clear what the natural smoothing hierarchy is. It is somewhat surprising that PDIA learning works nearly as well as $n$-gram modeling even without a smoothing hierarchy for its emission distributions. Imposing a hierarchical smoothing of the PDIA emission distributions remains an open problem.

## Footnotes

[1]PNFAs with no final probability are equivalent to hidden Markov models [3]

[2]In general $q_0$ may be replaced by a distribution over initial states.

# References

[1] R. Carrasco and J. Oncina. Learning stochastic regular grammars by means of a state merging method. *Grammatical Inference and Applications*, pages 139–152, 1994.

[2] L. Carroll. *Alice's Adventures in Wonderland*. Macmillan, 1865. URL http://www.gutenberg.org/etext/11.

[3] P. Dupont, F. Denis, and Y. Esposito. Links between probabilistic automata and hidden Markov models: probability distributions, learning models and induction algorithms. *Pattern recognition*, 38(9):1349–1371, 2005.

[4] J. Feldman and J.F. Hanna. The structure of responses to a sequence of binary events. *Journal of Mathematical Psychology*, 3(2):371–387, 1966.

[5] J. Gasthaus, F. Wood, and Y. W. Teh. Lossless compression based on the Sequence Memoizer. In *Data Compression Conference 2010*, pages 337–345, 2010.

[6] A. Gelman, J. B. Carlin, H. S. Stern, and D. B. Rubin. *Bayesian data analysis*. Chapman & Hall, New York, 1995.

[7] D. J. C. MacKay and L.C. Bauman Peto. A hierarchical Dirichlet language model. *Natural language engineering*, 1(2):289–307, 1995.

[8] K. Murphy. Hidden Markov model (HMM) toolbox for Matlab, 2005. URL http://www.cs.ubc.ca/ murphyk/Software/HMM/hmm.html.

[9] M.O. Rabin. Probabilistic automata. *Information and control*, 6(3):230–245, 1963.

[10] L. Rabiner. A tutorial on hidden Markov models and selected applications in speech recognition. *Proceedings of the IEEE*, 77:257–286, 1989.

[11] A.S. Reber. Implicit learning of artificial grammars. *Journal of verbal learning and verbal behavior*, 6 (6):855–863, 1967.

[12] D. Ron, Y. Singer, and N. Tishby. The power of amnesia: Learning probabilistic automata with variable memory length. *Machine learning*, 25(2):117–149, 1996.

[13] C.R. Shalizi and K.L. Shalizi. Blind construction of optimal nonlinear recursive predictors for discrete sequences. In *Proceedings of the 20th conference on Uncertainty in Artificial Intelligence*, pages 504–511. UAI Press, 2004.

[14] Y. W. Teh. A hierarchical Bayesian language model based on Pitman-Yor processes. In *Proceedings of the Association for Computational Linguistics*, pages 985–992, 2006.

[15] Y. W. Teh, M. I. Jordan, M. J. Beal, and D. M. Blei. Hierarchical Dirichlet processes. *Journal of the American Statistical Association*, 101(476):1566–1581, 2006.

[16] F. Thollard. Improving probabilistic grammatical inference core algorithms with post-processing techniques. In *Eighteenth International Conference on Machine Learning*, pages 561–568, 2001.

[17] F. Thollard, P. Dupont, and C. del la Higuera. Probabilistic DFA inference using Kullback-Leibler divergence and minimality. In *Seventeenth International Conference on Machine Learning*, pages 975–982. Citeseer, 2000.

[18] F. Wood, C. Archambeau, J. Gasthaus, L. James, and Y. W. Teh. A stochastic memoizer for sequence data. In *Proceedings of the 26th International Conference on Machine Learning*, pages 1129–1136, Montreal, Canada, 2009.

